# Active Exploration in Dynamic Environments

**Sebastian B. Thrun**
School of Computer Science
Carnegie Mellon University
Pittsburgh, PA 15213
E-mail: thrun@cs.cmu.edu

**Knut Möller**
University of Bonn
Dept. of Computer Science
Römerstr. 164
D-5300 Bonn, Germany

## Abstract

Whenever an agent learns to control an unknown environment, two opposing principles have to be combined, namely: *exploration* (long-term optimization) and *exploitation* (short-term optimization). Many real-valued connectionist approaches to learning control realize exploration by randomness in action selection. This might be disadvantageous when costs are assigned to "negative experiences". The basic idea presented in this paper is to make an agent explore unknown regions in a more directed manner. This is achieved by a so-called *competence map*, which is trained to predict the controller's accuracy, and is used for guiding exploration. Based on this, a bistable system enables smoothly switching attention between two behaviors – exploration and exploitation – depending on expected costs and knowledge gain.
The appropriateness of this method is demonstrated by a simple robot navigation task.

## INTRODUCTION

The need for exploration in adaptive control has been recognized by various authors [MB89, Sut90, Moo90, Sch90, BBS91]. Many connectionist approaches (e.g. [Mel89, MB89]) distinguish a *random exploration phase*, at which a controller is constructed by generating actions randomly, and a subsequent *exploitation phase*. Random exploration usually suffers from three major disadvantages:

- Whenever *costs* are assigned to certain experiences – which is the case for various real-world tasks such as autonomous robot learning, chemical control. flight control etc. –, exploration may become unnecessarily expensive. Intuitively speaking, a child would burn itself again and again simply because it is

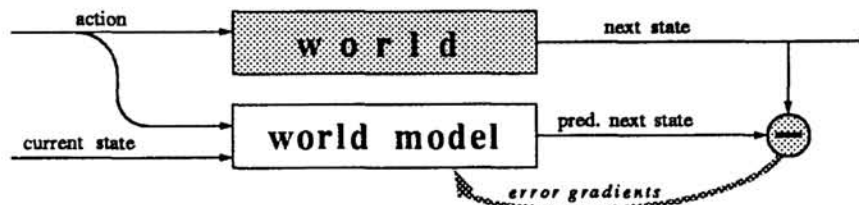

**Figure 1**: The training of the model network is a system identification task. Weights and biases of the network are estimated by gradient descent using the backpropagation algorithm.

in its random phase.

- Random exploration is often inefficient in terms of learning time, too [Whi91, Thr92]. Random actions usually make an agent waste plenty of time in already well-explored regions in state space, while other regions may still be poorly explored. Exploration happens by chance and is thus *undirected*.

- Once the exploitation phase begins, learning is finished and the system is unable to adapt to time-varying, dynamic environments.

However, more efficient exploration techniques rely on knowledge about the learning process itself, which is used for guiding exploration. Rather than selecting actions randomly, these exploration techniques select actions such that the expected knowledge gain is maximal. In discrete domains, this may be achieved by preferring states (or state-action pairs) that have been visited less frequently [BS90], or less recently [Sut90], or have previously shown a high prediction error [Moo90, Sch91][1]. For various discrete deterministic domains such exploration heuristics have been proved to prevent from exponential learning time [Thr92] (exponential in size of the state space). However, such techniques require a variable associated with each state-action pair, which is not feasible if states and actions are real-valued.

A novel real-valued generalization of these approaches is presented in this paper. A so-called *competence map* estimates the controller's accuracy. Using this estimation, the agent is driven into regions in state space with low accuracy, where the resulting learning effect is assumed to be maximal. This technique defines a *directed* exploration rule. In order to minimize costs during learning, exploration is combined with an exploitation mechanism using selective attention, which allows for switching between exploration and exploitation.

## INDIRECT CONTROL USING FORWARD MODELS

In this paper we focus on an adaptive control scheme adopted from Jordan [Jor89]:

**System identification** (Fig. 1): Observing the input-output behavior of the unknown world (environment), a model is constructed by minimizing the difference of the observed outcome and its corresponding predictions. This is done with backpropagation.

**Action search using the model network** (Fig. 2): Let an actual state $s$ and a goal state $s^*$ be given. Optimal actions are searched using gradient descent in action space: starting with an initial action (e.g. randomly chosen), the next state

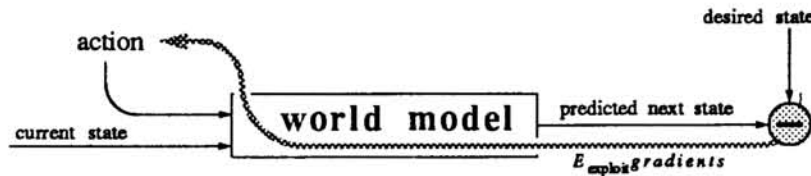

**Figure 2**: Using the model for optimizing actions (exploitation). Starting with some initial action, gradient descent through the model network progressively improves actions.

$\hat{s}$ is predicted with the world model. The *exploitation energy function*

$$E_{\text{exploit}} \;=\; (s^* - \hat{s})^T (s^* - \hat{s})$$

measures the LMS-deviation of the predicted and the desired state. Since the model network is differentiable, gradients of $E_{\text{exploit}}$ can be propagated back through the model network. Using these gradients, actions are optimized progressively by gradient descent in action space, minimizing $E_{\text{exploit}}$. The resulting actions *exploit* the world.

## THE COMPETENCE MAP

The general principle of many enhanced exploration schemes [BS90, Sut90, Moo90, TM91, Sch91, Thr92] is to select actions such that the resulting observations are expected to optimally improve the controller. In terms of the above control scheme, this may be realized by driving the agent into regions in state-action space where the accuracy of the model network is assumed to be low, and thus the knowledge gain by visiting these regions is assumed to be high. In order to estimate the accuracy of the model network, we introduce the notion of a *competence network* [Sch91, TM91]. Basically, this map estimates some upper bound of the LMS-error of the model network. This estimation is used for exploring the world by selecting actions which minimize the expected *competence* of the model, and thus maximize the resulting learning effect.

However, training the competence map is not as straightforward, since it is impossible to exactly predict the accuracy of the model network for regions in state space not visited for some time. The training procedure for the competence map is based on the assumption that the error increases (and thus competence decreases) slowly for such regions due to relearning and environmental dynamics:

1. At each time tick, backpropagation learning is applied using the last state-action pair as input, and the observed LMS-prediction error of the model as target value (c.f. Fig. 3), normalized to $(0, \varepsilon_{\text{max}})$ ($0 \leq \varepsilon_{\text{max}} \leq 1$, so far we used $\varepsilon_{\text{max}} = 1$).

2. For some[2] randomly generated state-action pairs, the competence map is subsequently trained with target 1.0 ($\leq$ largest possible error $\varepsilon_{\text{max}}$) [ACL$^+$90]. This training step establishes a heuristic, realizing the loss of accuracy in unvisited regions: over time, the output values of the competence map increase for these regions.

Actions are now selected with respect to an energy function $E$ which combines both

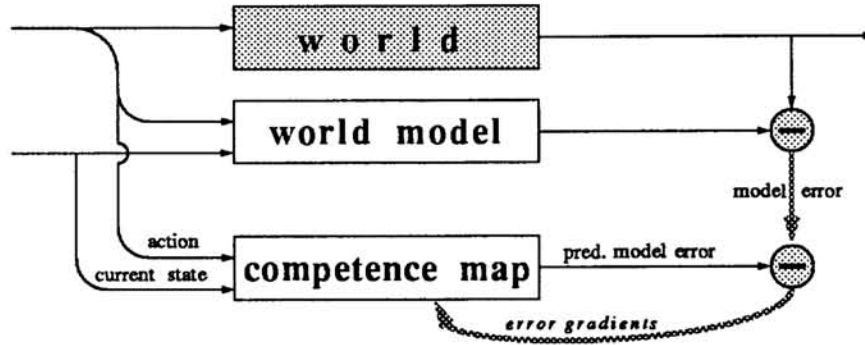

**Figure 3**: Training the competence map to predict the error of the model by gradient descent (see text).

exploration and exploitation:

$$E \;=\; (1-\Gamma)\cdot E_{\text{explore}} \;+\; \Gamma\cdot E_{\text{exploit}} \qquad (1)$$

with gain parameter $\Gamma$ ($0<\Gamma<1$). Here the *exploration energy*

$$E_{\text{explore}} \;=\; 1 - competence(action)$$

is evaluated using the competence map – minimizing $E_{\text{explore}}$ is equivalent to maximizing the predicted model error. Since both the model net and the competence net are differentiable, gradient descent in action space may be used for minimizing Eq. (1). $E$ combines exploration with exploitation: on the one hand minimizing $E_{\text{exploit}}$ serves to avoid costs (short-term optimization), and on the other hand minimizing $E_{\text{explore}}$ ensures exploration (long-term optimization). $\Gamma$ determines the portion of both target functions – which can be viewed to represent behaviors – in the action selection process.

Note that $\varepsilon_{\text{max}}$ determines the character of exploration: if $\varepsilon_{\text{max}}$ is large, the agent is attracted by regions in state space which have previously shown *high prediction error*. The smaller $\varepsilon_{\text{max}}$ is, the more the agent is attracted by *rarely-visited* regions.

## EXPLORATION AND SELECTIVE ATTENTION

Clearly, exploration and exploitation are often conflicting and can hinder each other. E.g. if exploration and exploitation pull a mobile robot into opposite directions, the system will stay where it is. It therefore makes sense not to keep $\Gamma$ constant during learning, but sometimes to focus more on exploration and sometimes more on exploitation, depending on expected costs and improvements. In our approach, this is achieved by determining the *focus of attention* $\Gamma$ using the following bistable recursive function which allows for smoothly switching attention between both policies. At each step of action search, let $e_{\text{exploit}} = \Delta E_{\text{exploit}}(a)$ and $e_{\text{explore}} = \Delta E_{\text{explore}}(a)$ denote the expected change of both energy functions by action $a$. With $f(\cdot)$ being a positive and monotonically increasing function[3],

$$\kappa \;\longleftarrow\; \Gamma\cdot f(e_{\text{exploit}}) \;-\; (1-\Gamma)\cdot f(e_{\text{explore}}) \qquad (2)$$

compares the influence of action $a$ on both energy functions *under the current focus of attention* $\Gamma$. The new $\Gamma$ is then derived by squashing $\kappa$ (with $c>0$):

$$\Gamma \;\longleftarrow\; \frac{1}{1+e^{-c\cdot\kappa}} \qquad (3)$$

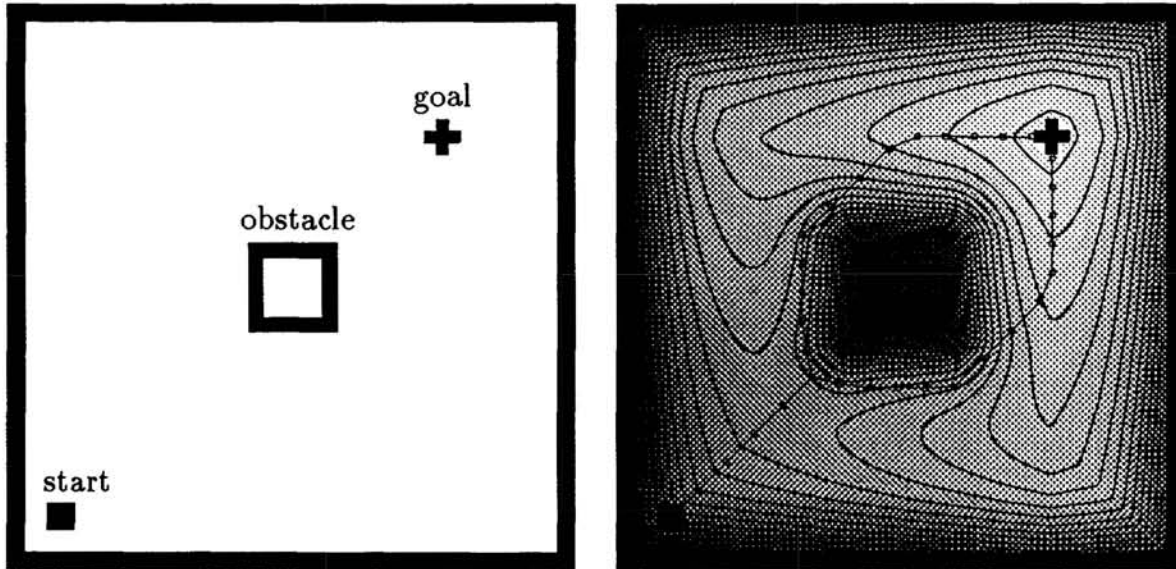

Figure 4: (a) Robot world – note that there are two equally good paths leading around the obstacle. (b) Potential field: In addition to the $x$-$y$-state vector, the environment returns for each state a *potential field value* (the darker the color, the larger the value). Gradient ascent in the potential field yields both optimal paths depicted. Learning this potential field function is part of the system identification task.

If $\kappa > 0$, the learning system is in exploitation mood and $\Gamma > 0.5$. Likewise, if $\kappa < 0$, the system is in exploration mood and $\Gamma < 0.5$. Since the actual attention $\Gamma$ weighs both competing energy functions, in most cases Eqs. (2) and (3) establish two stable points (fixpoints), close to 0 and 1, respectively. Attention is switched only if $\kappa$ changes its sign. The scalar $c$ serves as *stability factor*: the larger $c$ is, the closer is $\Gamma$ to its extremal values and the larger the switching factors $\Gamma(1-\Gamma)^{-1}$ (taken from Eq. (2)).

## A ROBOT NAVIGATION TASK

We now will demonstrate the benefits of active exploration using a competence map with selective attention by a simple robot navigation example. The environment is a 2-dimensional room with one obstacle and walls (see Fig. 4a), and $x$-$y$-states are evaluated by a potential field function (Fig. 4b). The goal is to navigate the robot from the start to the goal position without colliding with the obstacle or a wall.

Using a model network without hidden units for state prediction and a model with two hidden layers (10 units with gaussian activation functions in the first hidden layer, and 8 logistic units in the second) for potential field value prediction, we compared the following exploration techniques – Table 1 summarizes the results:

- **Pure random exploration.** In Fig. 5a the best result out of 20 runs is shown. The dark color in the middle indicates that the obstacle was touched extremely often. Moreover, the resulting controller (exploitation phase) did not find a path to the goal.

- **Pure exploitation** (see Fig. 5b). (With a bit of randomness in the beginning) this exploration technique found one of two paths but failed in both finding the other path and performing proper system identification. The number of crashes

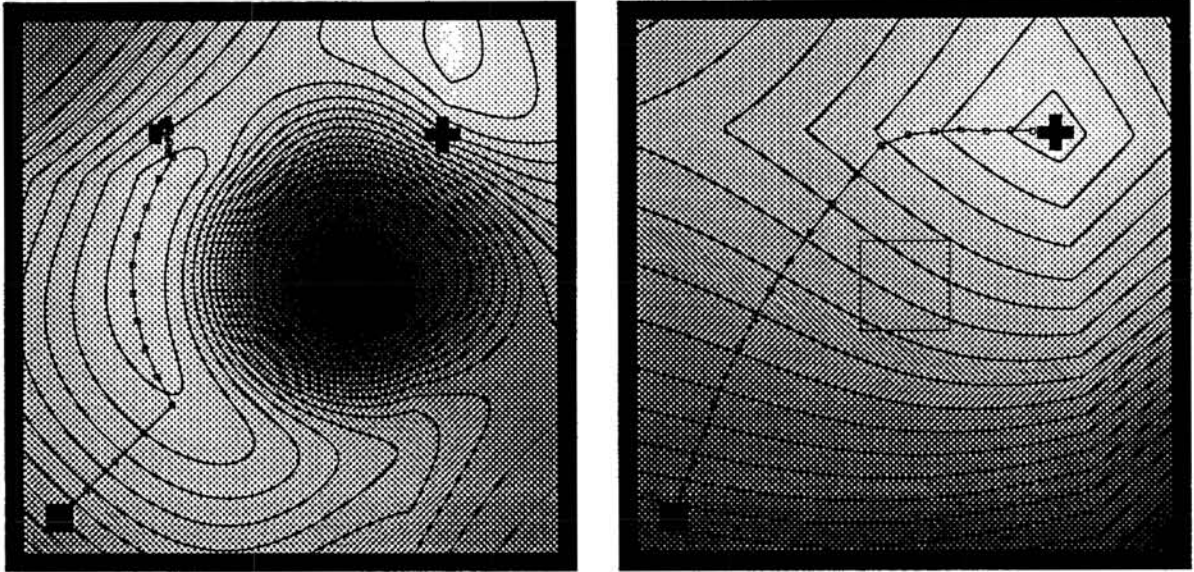

Figure 5: Resulting models of the potential field function. (a) **Random exploration.** The dark color in the middle indicates the high number of crashes against the obstacle. Note that the agent is restarted whenever it crashes against a wall or the obstacle – the probability for reaching the goal is 0.0007. (b) **Pure exploitation**: The resulting model is accurate along the path, but inaccurate elsewhere. Only one of two paths is identified.

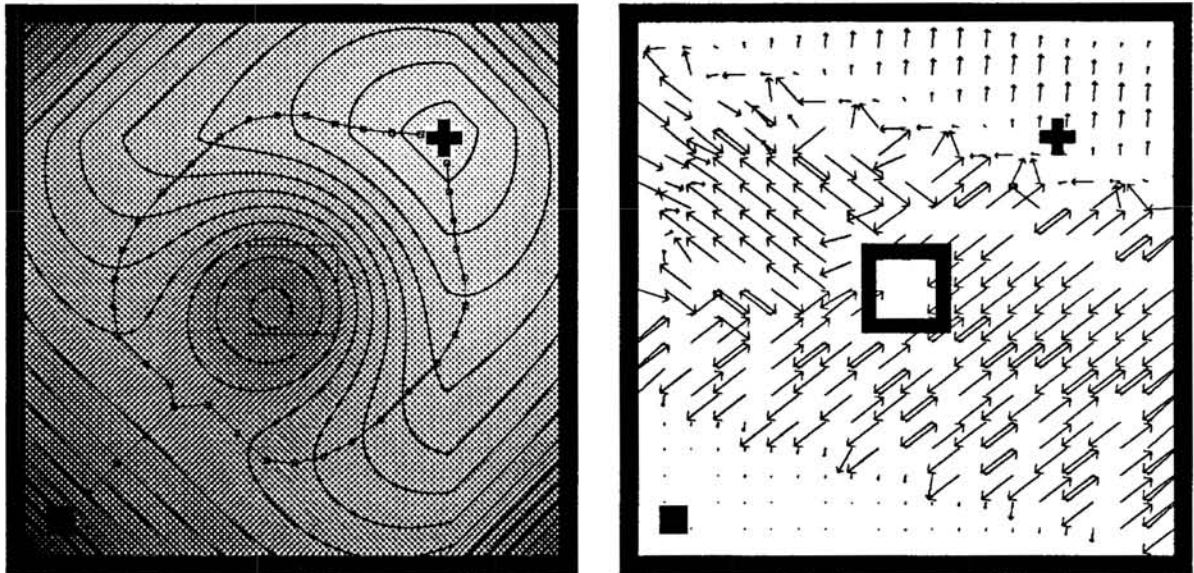

Figure 6: **Active exploration.** (a) Resulting model of the potential field function. This model is most accurate, and the number of crashes during training is the smallest. Both paths are found about equally often. (b) "Typical" competence map: The arrows indicate actions which maximize $E_{\text{explore}}$ (pure exploration).

|  | # runs | # crashes | # paths found | $L_2$-model error |
|---|---|---|---|---|
| random exploration | 10 000 | 9 993 | 0 | 2.5 % |
| pure exploitation | 15 000 | 11 000 | 1 | 0.7 % |
| active exploration | 15 000 | 4 000 | 2 | 0.4 % |

Table 1: Results (averaged over 20 runs). The $L_2$-model error is measured in relation to its initial value ($= 100\%$).

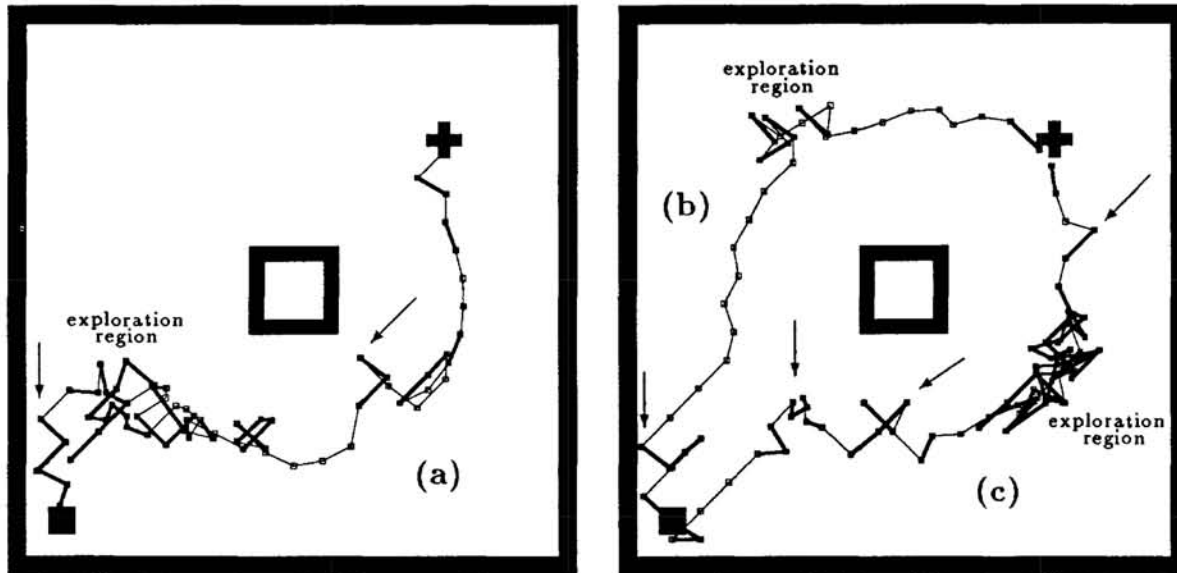

**Figure 7:** Three examples of trajectories during learning demonstrate the switching attention mechanism described in the paper. Thick lines indicate exploration mode ($\Gamma < 0.2$), and thin lines indicate exploitation ($\Gamma > 0.8$). The arrows mark some points where exploration is switched off due to a predicted collision.

during learning was significantly smaller than with random exploration.

* **Directed exploration with selective attention.** Using a competence network with two hidden layers (6 units each hidden layer), a proper model was found in all simulations we performed (Fig. 6a), and the number of collisions were the least. An intermediate state of the competence map is depicted in Fig. 6b, and three exploration runs are shown in Fig. 7.

## DISCUSSION

We have presented an adaptive strategy for efficient exploration in non-discrete environments. A so-called competence map is trained to estimate the competence (error) of the world model, and is used for driving the agent to less familiar regions. In order to avoid unnecessary exploration costs, a selective attention mechanism switches between exploration and exploitation. The resulting learning system is dynamic in the sense that whenever one particular region in state space is preferred for several runs, sooner or later the exploration behavior forces the agent to leave this region. Benefits of this exploration technique have been demonstrated on a robot navigation task.

However, it should be noted that the exploration method presented seeks to explore more or less the *whole* state-action space. This may be reasonable for the above robot navigation task, but many state spaces, e.g. those typically found in traditional AI, are too large for getting exhaustively explored even once. In order to deal with such spaces, this method should be extended by some mechanism for cutting off exploration in "unrelevant" regions in state-action space, which may be determined by some notion of "relevance".

Note that the technique presented here does not depend on the particular control scheme at hand. E.g., some exploration techniques in the context of reinforcement

learning may be found in [Sut90, BBS91], and are surveyed and compared in [Thr92].

## Acknowledgements

The authors wish to thank Jonathan Bachrach, Andy Barto, Jörg Kindermann, Long-Ji Lin, Alexander Linden, Tom Mitchell, Andy Moore, Satinder Singh, Don Sofge, Alex Waibel, and the reinforcement learning group at CMU for interesting and fruitful discussions. S. Thrun gratefully acknowledges the support by German National Research Center for Computer Science (GMD) where part of the research was done, and also the financial support from Siemens Corp.

## Footnotes

[1]Note that these two approaches [Moo90, Sch91] are real-valued.

[2]in our simulations: five – with a small learning rate

[3]We chosed $f(x) = e^x$ in our simulations.

## References

[ACL+90]  L. Atlas, D. Cohn, R. Ladner, M.A. El-Sharkawi, R.J. Marks, M.E. Aggoune, and D.C. Park. Training connectionist networks with queries and selective sampling. In D. Touretzky (ed.) *Advances in Neural Information Processing Systems 2*, San Mateo, CA, 1990. IEEE, Morgan Kaufmann.

[BBS91]  A.G. Barto, S.J. Bradtke, and S.P. Singh. Real-time learning and control using asynchronous dynamic programming. Technical Report COINS 91-57, Department of Computer Science, University of Massachusetts, MA, Aug. 1991.

[BS90]  A.G. Barto and S.P. Singh. On the computational economics of reinforcement learning. In D.S. Touretzky et al. (eds.), *Connectionist Models, Proceedings of the 1990 Summer School*, San Mateo, CA, 1990. Morgan Kaufmann.

[Jor89]  M.I. Jordan. Generic constraints on underspecified target trajectories. In *Proceedings of the First International Joint Conference on Neural Networks, Washington, DC*, IEEE TAB Neural Network Committee, San Diego, 1989.

[MB89]  M.C. Mozer and J.R. Bachrach. Discovering the structure of a reactive environment by exploration. Technical Report CU-CS-451-89, Dept. of Computer Science, University of Colorado, Boulder, Nov. 1989.

[Mel89]  B.W. Mel. Murphy: A neurally-inspired connectionist approach to learning and performance in vision-based robot motion planning. Technical Report CCSR-89-17A, Center for Complex Systems Research Beckman Institute, University of Illinois, 1989.

[Moo90]  A.W. Moore. *Efficient Memory-based Learning for Robot Control*. PhD thesis, Trinity Hall, University of Cambridge, England, 1990.

[Sch90]  J.H. Schmidhuber. Making the world differentiable: On using supervised learning fully recurrent neural networks for dynamic reinforcement learning and planning in non-stationary environments. Technical Report, Technische Universität München, Germany, 1990.

[Sch91]  J.H. Schmidhuber. Adaptive confidence and adaptive curiosity. Technical Report FKI-149-91, Technische Universität München, Germany 1991.

[Sut90]  R.S. Sutton. Integrated architectures for learning, planning, and reacting based on approximating dynamic programming. In *Proceedings of the Seventh International Conference on Machine Learning*, June 1990.

[TM91]  S.B. Thrun and K. Möller. On planning and exploration in non-discrete environments. Technical Report 528, GMD, St.Augustin, FRG, 1991.

[Thr92]  S.B. Thrun. Efficient exploration in reinforcement learning. Technical Report CMU-CS-92-102, Carnegie Mellon University, Pittsburgh, Jan. 1992.

[Whi91]  S.D. Whitehead. A study of cooperative mechanisms for faster reinforcement learning. Technical Report 365, University of Rochester, Computer Science Department, Rochester, NY, March 1991.
